# Responding to modalities with different latencies

**Fredrik Bissmarck**
Computational Neuroscience Labs
ATR International
Hikari-dai 2-2-2, Seika, Soraku
Kyoto 619-0288 JAPAN
xfredrik@atr.jp

**Hiroyuki Nakahara**
Laboratory for Mathematical Neuroscience
RIKEN Brain Science Institute
Hirosawa 2-1-1, Wako
Saitama 351-0198 JAPAN
hiro@brain.riken.jp

**Kenji Doya**
Initial Research Project
Okinawa Institute of Science and Technology
12-22 Suzaki, Gushikawa
Okinawa 904-2234 JAPAN
doya@irp.oist.jp

**Okihide Hikosaka**
Laboratory of Sensorimotor Research
National Eye Institute, NIH
Building 49, Room 2A50
Bethesda, MD 20892
oh@lsr.nei.nih.gov

## Abstract

Motor control depends on sensory feedback in multiple modalities with different latencies. In this paper we consider within the framework of reinforcement learning how different sensory modalities can be combined and selected for real-time, optimal movement control. We propose an actor-critic architecture with multiple modules, whose output are combined using a softmax function. We tested our architecture in a simulation of a sequential reaching task. Reaching was initially guided by visual feedback with a long latency. Our learning scheme allowed the agent to utilize the somatosensory feedback with shorter latency when the hand is near the experienced trajectory. In simulations with different latencies for visual and somatosensory feedback, we found that the agent depended more on feedback with shorter latency.

## 1 Introduction

For motor response, the brain relies on several modalities. These may carry different information. For example, vision keeps us updated on external world events, while somatosensation gives us detailed information about the state of the motor system. For most human behaviour, both are crucial for optimal performance.

However, modalities may also differ in latency. For example, information may be perceived faster by the somatosensory pathway than the visual. For quick responses it would be reasonable that the modality with shorter latency is more important. The slower modality would be useful if it carries additional information, for example when we have to attend to a visual cue.

There has been a lot of research on modular organisation where each module is an expert of a particular part of the state space (e.g. [1]). We address questions concerning modules with

different feedback delays, and how they are used for real-time motor control. How does the latency affect the influence of a modality over action? How can modalities be combined? Here, we propose an actor-critic framework, where modules compete for influence over action by reinforcement. First, we present the generic framework and learning algorithm. Then, we apply our model to a visuomotor sequence learning task, and give details of the simulation results.

## 2 General framework

This section describes the generic concepts of our model: a set of modules with delayed feedback, a function for combining them and a learning algorithm.

### 2.1 Network architecture

Consider $M$ modules, where each module has its own feedback signal $\mathbf{y}^m(\mathbf{x}(t-\tau^m))$ $(m = 1, 2, .., M)$ computed from the state of the environment $\mathbf{x}(t)$. Each module has a corresponding time delay $\tau^m$ (see figure 1). (The same feedback signals are used to compute the critic, see the next subsection). Each module outputs a population-coded output $\mathbf{a}^m(t)$, where each element $a_j^m$ $(j = 1, 2, ..J)$ corresponds to the motor output vector $\mathbf{u}_j$, which represents, for example, joint torques. The output of an actor is given by a function approximator $\mathbf{a}^m(t) = \mathrm{f}(\mathbf{y}^m(t-\tau^m); \mathbf{w}^m)$ with parameters $\mathbf{w}^m$.

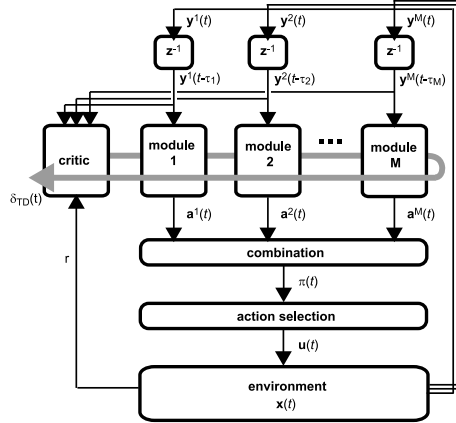

Figure 1: The general framework.

The actual motor command $\mathbf{u} \in R^D$ is given by combination of population vector outputs $\mathbf{a}^m$ of the modules. Here we consider the use of softmax combination. The probablity of taking $j$-th motor output vector is given by

$$\pi_j(t) = \frac{\exp\left(\beta\sum_{m=1}^M a_j^m\right)}{\sum_{j=1}^J \exp\left(\beta\sum_{m=1}^M a_j^m\right)},$$

where $\beta$ is the inverse temperature, controlling the stochasticity. At each moment, one of the motor command vectors is selected as $p(\mathbf{u}(t) = \bar{\mathbf{u}}_j) = \pi_j(t)$. We define $\mathbf{q}(t)$ to be a binary vector of J elements where the one corresponding to the chosen action is 1 and others 0.

There is no explicit mechanism in the architecture that explicitly favour a module with shorter latency. Instead, we test whether a reinforcement learning algorithm can learn to select the modules which are more useful to the agent.

### 2.2 Learning algorithm

Our model is a form of the continuous actor-critic [2]. The function of the critic is to estimate the expected future reward, i.e. to learn the value function

$$V = V(\mathbf{y}^1(t - \tau^1), \mathbf{y}^2(t - \tau^2), .., \mathbf{y}^M(t - \tau^M); \mathbf{w}^c)$$

where $\mathbf{w}^c$ is a set of trainable parameters. The temporal difference (TD) error $\delta^{TD}$ is the discrepancy between expected and actual reward $r(t)$. In its continuous form:

$$\delta^{TD}(t) = r(t) - \frac{1}{\tau^{TD}}V(t) + \dot{V}(t)$$

where $\tau^{TD}$ is the future reward discount time constant.

The TD error is used to update the parameters for both the critic, and the actor, which in our framework is the set of modules.

Learning of each actor module is guided by the action deviation signal

$$E_j(t) = \frac{(q_j(t) - \pi_j(t))^2}{2}$$

which is the difference between the its output and the action that was actually selected.

Parameters of the critic and actors are updated using eligibility traces

$$\dot{e}^c_k(t) = -\frac{1}{\kappa}e^c_k + \frac{\partial V}{\partial w^c_k} \qquad \dot{e}^m_{kj}(t) = -\frac{1}{\kappa}e^c_{kj} + \frac{\partial E_j(t)}{\partial w^m_{kj}}$$

where $k$ is the index of parameters and $\kappa$ is a time constant. The trace for $m$-th actor is given from

$$\frac{\partial E_j(t)}{\partial w^m_{kj}} = (q_j(t) - \pi_j(t))\frac{\partial \pi_j(t)}{\partial w^m_{kj}}$$

The parameters are updated by gradient descent as

$$\dot{w}^c_k = \alpha\delta^{TD}(t)e^c_k(t) \qquad \dot{w}^m_{kj} = \alpha\delta^{TD}(t)e^m_{kj}(t)$$

where $\alpha$ denotes the learning rate.

### 2.3 Neuroanatomical correlates

Our network architecture is modeled to resemble the function of the basal ganglia-thalamocortical (BG-TC) system to select and learn actions for goal-directed movement. Actor-critic models of the basal ganglia has been proposed by many (e.g. [3], [4]). The modular organisation of the BG-TC loop circuits ([5], [6]), where modules depends on different sensory feedback, implies that the actor-critic depends on several modules.

## 3 An example application

To demonstrate our paradigm we exemplify by a motor sequence learning task, inspired by "the n x n task", an experimental paradigm where monkeys and humans learn a sequence of reaching movements, where error performance improved across days, and performance time decreased across months [7]. The results from these experiments suggested that the influence of the motor BG-TC loop for motor execution is relatively stronger for learned sequences than for new ones, compared to the prefrontal BG-TC loop. In our model implementation, we want to investigate how the feedback delay affects the influence of visual and sensorimotor modalities when learning a stereotype real-time motor sequence. In our implementation (see figure 2), we use two modules, one "visual", and one "motor", corresponding to visual and somatosensory feedback respectively. The visual module represents

a preknown, visually guided reaching policy for arbitrary start and endpoints within reach. This module does not learn. The motor module represents the motor skill memory to be learned. It gives zero output initially, but learn by associating reinforcement with sequences of actions. The controlled object is a 2DOF arm, for which the agent gives a joint torque motor command, with action selection sampled at 100 Hz.

## 3.1 Environment

The environment consists of a 2DOF arm (both links are 0.3 m long and 0.1 m in diam., weight 1.0 kg), starting at position $S$, directly controlled by the agent, and a variable target (see environment box in figure 2). The task is to press three targets in consecutive order, which always appear at the same positions (one at one time), marked 1, 2 and 3 in the figure. If the hand of the arm satisfy a proximity condition ($|\boldsymbol{\xi}^{target} - \boldsymbol{\xi}^{hand}| < \xi^{prox}$ and $|\dot{\boldsymbol{\xi}}^{hand}| < v^{prox}$) a key (target) is considered pressed, and the next target appears immediately. To allow a larger possibility of modifying the movement, we have a very loose velocity constraint $v^{prox}$ (for all simulations, $\xi^{prox} = 0.02$ m and $v^{prox} = 0.5$ m/s ). Each trial ended after successful completion of the task, or after 5 s.

For each successful key press, the agent is rewarded instantaneously, with an increasing amount of reward for later keys in the sequence (50, 100, 150 respectively). A small, constant running time cost (10/s) was subtracted from the reward function $r(t)$.

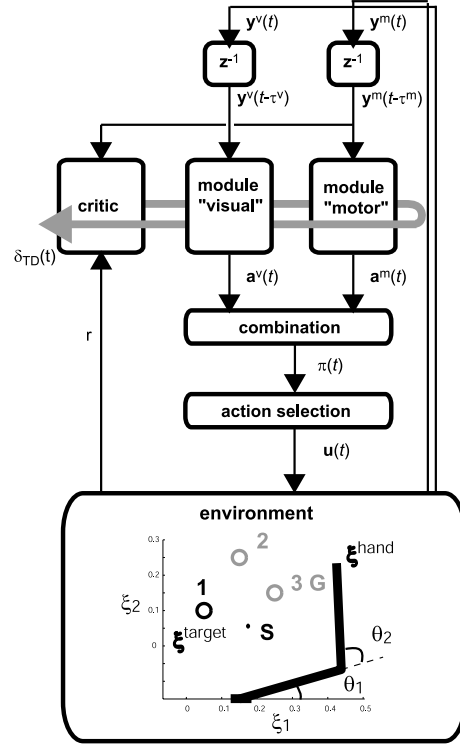

Figure 2: Implementation of the example simulation. The visual module is fed back the hand position $\{\xi_1^{hand}, \xi_2^{hand}\}$ and the position of the active target $\{\xi_1^{target}, \xi_2^{target}\}$, while the motor module is fed back a population code representing the joint angles $\{\theta_1, \theta_2\}$. S : Start, G : Goal. See text for further details.

## 3.2 The visual module

The visual module is designed as a computed torque feedback controller for simplicity. It was designed to give an output as similar as possible to biological reaching movements, but we did not attempt to design the controller itself in a biologically plausible way.

The feedback signal $\mathbf{y}^v$ to the visual module consists of the hand kinematics $\boldsymbol{\xi}^{hand}$, $\dot{\boldsymbol{\xi}}^{hand}$ and the target position $\boldsymbol{\xi}^{target}$. Using a computed torque feedback control law, the visual module uses these signals to generate a reaching movement, representing the preknown motor behaviour of the agent. As such a control law does not have measures to deal with delayed signals, we make the assumption that the control law relies on $\tilde{\boldsymbol{\xi}}^{hand}(t) = \boldsymbol{\xi}^{hand}(t)$, i.e. the controller can predict for the delay regarding the arm

movement (the target signal is still delayed by $\tau^v$. This is a limitation of our example, but is a necessity to avoid "motor babbling ", for which learning time would be infinitely long.

The controller output

$$\dot{\mathbf{u}}^{visual}(t) = -\frac{1}{\tau^{CT}}\mathbf{u}^{visual}(t) + \lambda\mathbf{u}^{visual\prime}(\ddot{\tilde{\boldsymbol{\xi}}}^{hand}, \dot{\tilde{\boldsymbol{\xi}}}^{hand}, \mathbf{e})$$

where $\tau^{CT}$ and $\lambda$ are constants, $\mathbf{e} = \boldsymbol{\xi}^{target}(t - \tau^v) - \tilde{\boldsymbol{\xi}}^{hand}(t)$ and

$$\mathbf{u}^{visual\prime}(t) = \mathbf{J}^T(\mathbf{M}(\ddot{\tilde{\boldsymbol{\xi}}}^{hand} + \mathbf{K}_1\dot{\tilde{\boldsymbol{\xi}}}^{hand} - \mathbf{K}_2\mathbf{e}) + \mathbf{C}\dot{\tilde{\boldsymbol{\xi}}}^{hand})$$

where $\mathbf{J}$ is the Jacobian $(\partial\boldsymbol{\theta}/\partial\tilde{\boldsymbol{\xi}}^{hand})$, $\mathbf{M}$ the moment of inertia matrix and $\mathbf{C}$ the Coriolis matrix. With proper control gains $\mathbf{K}_1$ and $\mathbf{K}_2$, the filter helps to give bell-shaped velocity profiles for the reaching movement, desirable for its resemblance to biological motion.

The output $\mathbf{u}^{visual}$ is then expanded to a population vector

$$a_j^v(t) = \frac{1}{Z}\exp(-\frac{1}{2}\{\sum_d(\frac{u_d^{visual}(t) - \bar{u}_{jd}}{\sigma''_{jd}})^2\})$$

where $Z$ is the normalisation term, $\bar{u}_{jd}$ is a preferable joint torque for Cartesian dimension $d$ for vector element $j$, $\sigma''_{jd}$ the corresponding variance.

**Parameters**: $\tau^{CT} = 50$ ms, $\lambda = 100$, $\mathbf{K}_1 = [10\ 0;0\ 10]$, $\mathbf{K}_2 = [50\ 0;0\ 50]$. The prefered joint torques $\bar{\mathbf{u}}_j$ corresponding to action $j$ were distributed symmetrically over the origin in a 5x5 grid, in the range (-100:100,-100:100) with the middle (0,0) unit removed. The corresponding variances $\sigma''_{jd}$ were half the distance to the closest node in each direction.

### 3.3 The motor module

The motor module relies on information about the motor state of the arm. In the vicinity of a target, by the immediate motor state alone it may be difficult to determine whether the hand should move towards or away from the target position. We solve this by adding contextual neurons. These neurons fire after a particular key is pressed.

Thus, the feedback signal $\mathbf{y}^m$ with $k = 1, 2, .., K$ is partitioned by $K_0$: The first part $(k \leq K_0)$ represents the motor state, and the second part $(k > K_0)$ represents the context.

The feedback to the motor module are the joint angles and angular velocities $\boldsymbol{\theta}$, $\dot{\boldsymbol{\theta}}$ of the arm, expanded to a population vector with $K_0$ elements:

$$y_k^m(t) = \frac{1}{Z}\exp\left(-\frac{1}{2}\{\sum_d(\frac{\theta_d(t) - \bar{\theta}_{kd}}{\sigma_{kd}})^2 + \sum_d(\frac{\dot{\theta}_d(t) - \bar{\omega}_{kd}}{\sigma'_{kd}})^2\}\right)$$

where $\bar{\theta}_{kd}$, $\bar{\omega}_{kd}$ are preferable joint angles and velocities, $\sigma_{kd}$ and $\sigma'_{kd}$ are corresponding variances, $Z$ is a normalisation term.

The context units are a number of $n = 1, 2, .., N$ tapped delay lines (where $N$ correspond to the number of keys in the sequence), where each delay line has $Q$ units. For $(k > K_0$, $k \neq K_0 + Q(n - 1) + 1)$:

$$\dot{y}_k^m(t) = -\frac{1}{\tau^C}y_k^m(t) + y_{k-1}(t)$$

Each delay line is initiated by the input at $(k = K_0 + Q(n - 1) + 1)$:

$$y_k^m(t) = \delta(t - \tau_n^{keypress})$$

where $\delta$ is the Dirac delta function, and $\tau_n^{keypress}$ is the instant the $n$th key was pressed.

The response signal $\mathbf{a}^m$ is the linear combination of $\mathbf{y}^m$ and the trainable matrix $\mathbf{W}^m$,

$$\mathbf{a}^m(t) = \mathbf{W}^m \mathbf{y}^m(t - \tau^m)$$

Though it is reasonable to use both feedback pathways for the critic, for simplicity we use only the motor:

$$V(t) = \mathbf{W}^c \mathbf{y}^m(t - \tau^m)$$

**Parameters**: The prefered joint angles $\bar{\theta}_{kd}$ and angular velocities $\bar{\omega}_{kd}$ were distributed uniformly in a 7*7*3*3 grid ($K_0 = 441$ nodes) for $k = 1, 2, ..K_0$ nodes , in the ranges (-0.2:1.2,1,2:1.6) rad and (-1:1,-1:1) rad/s. The corresponding variances $\sigma^{kd}$ and $\sigma'_{kd}$ were half the distance to the closest node in each direction. The contextual part of the vector has $Q = 8$, $N = 3$, which makes 24 elements. The time constant $\tau^C = 30$ ms.

## 4 Simulation results

We trained the model for four different feedback delay pairs ($\tau^v$ / $\tau^m$, in ms): 100/0, 100/50, 100/100, 0/100 ($\beta = 10$, $\tau^{TD} = 200$ ms, $\kappa = 200$ ms, $\alpha = 0.1$ s$^{-1}$). We stopped the simulations after 125,000 trials. Two properties are essential for our argument: the shortest feedback delay $\tau^{min} = \min(\tau^v, \tau^m)$ and the relative latency $\Delta\tau = (\tau^v - \tau^m)$.

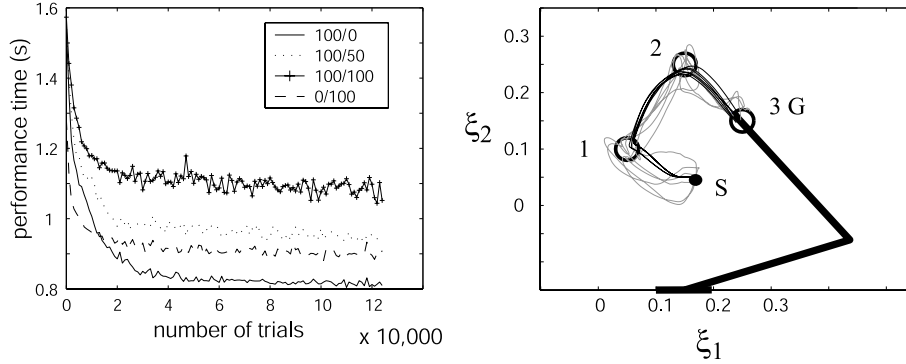

Figure 3: (Left) Change in performance time (running averages) across trials for different feedback delays (displayed in ms as visual/motor). (Right) Example hand trajectories for the initial (gray lines) and learned (black lines) behaviour for the run with 100 ms/0 ms delay.

### 4.1 Final performance time depends on the shortest latency

Figure 3 shows that the performance time (PT, the time it takes to complete one trial) was improved for all four simulations. The final PT relates to the shortest latency $\tau^{min}$, the shorter the better final performance.

However, there are three possible reasons for speedup: 1) a more deterministic (greedy) policy $\pi$, 2) a change in trajectory and 3) faster reaction by utilization of faster feedback. As we observed more stereotyped trajectories and more deterministic policies after learning, reason 1) is true, but does it account for the entire improvement? For the rather exploratory, visually guided initial movement, the average PT is $1.55\,\text{s}$ and $1.25\,\text{s}$ for $\tau^v = 100\,\text{ms}$ and $\tau^v = 0\,\text{ms}$ respectively, while the corresponding greedy policy PTs are $1.41\,\text{s}$ and $1.13\,\text{s}$. Since the final PTs always were lower, the speedup must also be due to other changes in behaviour. Figure 3 (right) shows example trajectories of the inital (gray) and learned (black) policy in 100/0. We see that while the initial movement was directed target-by-target, the learned displays a smoothly curved movement, optimized to perform the entire sequence. This is expected, as the discounted reward (determined by $\tau^{TD}$) and time cost favour fast movements over slow. This change was to some degree observed in all four simulations, although it was most evident (see the next subsection) in the 100/0. So reason 2) also seems to be true. We also see that the shorter $\tau^{min}$, the shorter final PT. Reason 3) is also significant: the possibility to speed up the movement is limited by $\tau^{min}$.

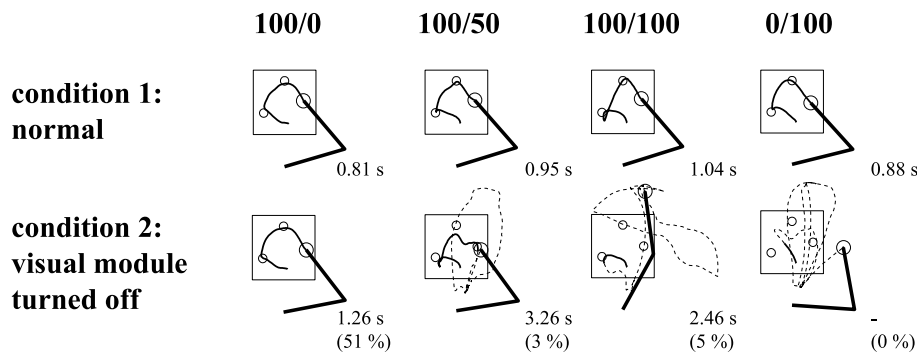

Figure 4: Performance after learning with typical examples of hand trajectories in a normal condition, and a condition with the visual module turned off, for agents with different feedback delay. Average performance times are displayed for each. When the visual module was turned off, the agent often failed to complete the sequence in 5 s. Success rate are shown in parantheses, and the corresponding average are for the successful trials only. The solid lines highlight the trajectory while execution is stable, while the dashed lines show the parts when the agent is out of control.

## 4.2 The module with shorter latency is more influential over motor control

Figure 4 shows the performance of sufficiently learned behaviour (after 125,000 trials) for two conditions: one normal ("condition 1") and one with the visual module turned off ("condition 2"). Condition 1 is shown mainly for reference. The difference in trajectories in condition 1 are marginal, but execution tends to destabilize with longer $\tau^{min}$. Condition 2 reveals the dependence of the visual module. In the 100/0 case, the correct spatial trajectory is generated each time, but a sometimes too fast movement leads to overshoots for 2nd and 3rd keys. For smaller $\Delta\tau$ (rightwards in figure 4) the execution becomes unstable, and the 0/100 case it could never execute the movement. For some reason, when the 100/100 kept the hand on track, it was less likely to do overshoots than the 100/50 case, which is why the average PT and success rate is better.

Thus, we conclude that the faster module are more influential over motor control. The adaptiveness of the motor loop also offer the motor module an advantage over the visual.

# 5   Conclusion

Our framework offers a natural way to combine modules with different feedback latencies. In any particular situation, the learning algorithm will reinforce the better module to use. When execution is fast, the module with shorter latency may be favourable, and when slow, the one with more information. For example, in vicinity of the experienced sequence, our agent utilized the somatosensory feedback to execute the movement more quickly, but once it lost control the visual feedback was needed to put the arm back on track again.

By using the softmax function it is possible to flexibly gate or combine module outputs. Sometimes the asynchrony of modules can cause the visual and motor modules to be directed towards different targets. Then it is desirable to suppress the slower module to favour the faster, which also occured in our example by reinforcing the motor module enough to suppress the visual. In other situations the reliability of one module may be insufficient for robust execution, making it necessary to combine modules.

In our 100/0 example, the slower visual module was used to assist the faster motor module to learn a skill. Once acquired, the visual module was not necessary for the skillful execution anymore, unless something went wrong. Thus, the visual module is more free to attend to other tasks. When we learn to ride a bicycle, for example, we first need to attend to what we do, but once we have learned, we can attend to other things, like the surrounding traffic or a conversation. Our result suggests that a longer relative latency helps to make the faster modality independent, so the slower can be decoupled from execution after learning.

In the human brain, forward models are likely to have access to an efference copy of the motor command, which may be more important than the incoming feedback for fast movements [1]. This is something we intend to look at in future work. Also, we will extend this work with a more theoretical analysis, and compare the performance of multiple adaptive modules.

## Acknowledgements

The research is supported by CREST. The authors would like to thank Erhan Oztop and Jun Morimoto for helpful comments.

## References

[1] M. Haruno, D. M. Wolpert, and M. Kawato. Mosaic model for sensorimotor learning and control. *Neural Comput*, 13(10):2201–20, 2001.

[2] K. Doya. Reinforcement learning in continuous time and space. *Neural Comput*, 12(1):219–45, 2000.

[3] K. Doya. What are the computations of the cerebellum, the basal ganglia and the cerebral cortex? *Neural Netw*, 12(7-8):961–974, 1999.

[4] N. Daw. *Reinforcement learning models of the dopamine system and their behavioral implications*. PhD thesis, Carnegie Mellon University, 2003.

[5] G. E. Alexander and M. D. Crutcher. Functional architecture of basal ganglia circuits: neural substrates of parallel processing. *Trends Neurosci*, 13(7):266–71, 1990.

[6] H. Nakahara, K. Doya, and O. Hikosaka. Parallel cortico-basal ganglia mechanisms for acquisition and execution of visuomotor sequences - a computational approach. *J Cogn Neurosci*, 13(5):626–47, 2001.

[7] O. Hikosaka, H. Nakahara, M. K. Rand, K. Sakai, X. Lu, K. Nakamura, S. Miyachi, and K. Doya. Parallel neural networks for learning sequential procedures. *Trends Neurosci*, 22(10):464–71, 1999.
